# Worst-Case Linear Discriminant Analysis

**Yu Zhang and Dit-Yan Yeung**
Department of Computer Science and Engineering
Hong Kong University of Science and Technology
{zhangyu,dyyeung}@cse.ust.hk

## Abstract

Dimensionality reduction is often needed in many applications due to the high dimensionality of the data involved. In this paper, we first analyze the scatter measures used in the conventional linear discriminant analysis (LDA) model and note that the formulation is based on the average-case view. Based on this analysis, we then propose a new dimensionality reduction method called worst-case linear discriminant analysis (WLDA) by defining new between-class and within-class scatter measures. This new model adopts the worst-case view which arguably is more suitable for applications such as classification. When the number of training data points or the number of features is not very large, we relax the optimization problem involved and formulate it as a metric learning problem. Otherwise, we take a greedy approach by finding one direction of the transformation at a time. Moreover, we also analyze a special case of WLDA to show its relationship with conventional LDA. Experiments conducted on several benchmark datasets demonstrate the effectiveness of WLDA when compared with some related dimensionality reduction methods.

## 1 Introduction

With the development of advanced data collection techniques, large quantities of high-dimensional data are commonly available in many applications. While high-dimensional data can bring us more information, processing and storing such data poses many challenges. From the machine learning perspective, we need a very large number of training data points to learn an accurate model due to the so-called 'curse of dimensionality'. To alleviate these problems, one common approach is to perform dimensionality reduction on the data. An assumption underlying many dimensionality reduction techniques is that the most useful information in many high-dimensional datasets resides in a low-dimensional latent space. Principal component analysis (PCA) [8] and linear discriminant analysis (LDA) [7] are two classical dimensionality reduction methods that are still widely used in many applications. PCA, as an unsupervised linear dimensionality reduction method, finds a low-dimensional subspace that preserves as much of the data variance as possible. On the other hand, LDA is a supervised linear dimensionality reduction method which seeks to find a low-dimensional subspace that keeps data points from different classes far apart and those from the same class as close as possible.

The focus of this paper is on the supervised dimensionality reduction setting like that for LDA. To set the stage, we first analyze the between-class and within-class scatter measures used in conventional LDA. We then establish that conventional LDA seeks to maximize the average pairwise distance between class means and minimize the average within-class pairwise distance over all classes. Note that if the purpose of applying LDA is to increase the accuracy of the subsequent classification task, then it is desirable for *every* pairwise distance between two class means to be as large as possible and *every* within-class pairwise distance to be as small as possible, but not just the *average* distances. To put this thinking into practice, we incorporate a worst-case view to define a new between-class

scatter measure as the minimum of the pairwise distances between class means, and a new within-class scatter measure as the maximum of the within-class pairwise distances over all classes. Based on the new scatter measures, we propose a novel dimensionality reduction method called worst-case linear discriminant analysis (WLDA). WLDA solves an optimization problem which simultaneously maximizes the worst-case between-class scatter measure and minimizes the worst-case within-class scatter measure. If the number of training data points or the number of features is not very large, e.g., below 100, we propose to relax the optimization problem and formulate it as a metric learning problem. In case both the number of training data points and the number of features are large, we propose a greedy approach based on the constrained concave-convex procedure (CCCP) [24, 18] to find one direction of the transformation at a time with the other directions fixed. Moreover, we also analyze a special case of WLDA to show its relationship with conventional LDA. We will report experiments conducted on several benchmark datasets.

## 2 Worst-Case Linear Discriminant Analysis

We are given a training set of $n$ data points, $\mathcal{D} = \{\mathbf{x}_1, \ldots, \mathbf{x}_n\} \subset \mathbb{R}^d$. Let $\mathcal{D}$ be partitioned into $C \geq 2$ disjoint classes $\Pi_i$, $i = 1, \ldots, C$, where class $\Pi_i$ contains $n_i$ examples. We perform linear dimensionality reduction by finding a transformation matrix $\mathbf{W} \in \mathbb{R}^{d \times r}$.

### 2.1 Objective Function

We first briefly review the conventional LDA. The between-class scatter matrix and within-class scatter matrix are defined as

$$\mathbf{S}_b = \sum_{k=1}^{C} \frac{n_k}{n} (\bar{\mathbf{m}}_k - \bar{\mathbf{m}})(\bar{\mathbf{m}}_k - \bar{\mathbf{m}})^T, \qquad \mathbf{S}_w = \sum_{k=1}^{C} \sum_{\mathbf{x}_i \in \Pi_k} \frac{1}{n} (\mathbf{x}_i - \bar{\mathbf{m}}_k)(\mathbf{x}_i - \bar{\mathbf{m}}_k)^T,$$

where $\bar{\mathbf{m}}_k = \frac{1}{n_k} \sum_{\mathbf{x}_i \in \Pi_k} \mathbf{x}_i$ is the class mean of the $k$th class $\Pi_k$ and $\bar{\mathbf{m}} = \frac{1}{n} \sum_{i=1}^{n} \mathbf{x}_i$ is the sample mean of all data points. Based on the scatter matrices, the between-class scatter measure and within-class scatter measure are defined as

$$\eta_b = \operatorname{tr}(\mathbf{W}^T \mathbf{S}_b \mathbf{W}), \qquad \eta_w = \operatorname{tr}(\mathbf{W}^T \mathbf{S}_w \mathbf{W}),$$

where $\operatorname{tr}(\cdot)$ denotes the trace of a square matrix. LDA seeks to find the optimal solution of $\mathbf{W}$ that maximizes the ratio $\eta_b / \eta_w$ as the optimality criterion.

By using the fact that $\bar{\mathbf{m}} = \frac{1}{n} \sum_{k=1}^{C} n_k \bar{\mathbf{m}}_k$, we can rewrite $\mathbf{S}_b$ as

$$\mathbf{S}_b = \frac{1}{2n^2} \sum_{i=1}^{C} \sum_{j=1}^{C} n_i n_j (\bar{\mathbf{m}}_i - \bar{\mathbf{m}}_j)(\bar{\mathbf{m}}_i - \bar{\mathbf{m}}_j)^T.$$

According to this and the definition of the within-class scatter measure, we can see that LDA tries to maximize the average pairwise distance between class means $\{\bar{\mathbf{m}}_i\}$ and minimize the average within-class pairwise distance over all classes. Instead of taking this average-case view, our WLDA model adopts a worst-case view which arguably is more suitable for classification applications.

We define the sample covariance matrix for the $k$th class $\Pi_k$ as

$$\mathbf{S}_k = \frac{1}{n_k} \sum_{\mathbf{x}_i \in \Pi_k} (\mathbf{x}_i - \bar{\mathbf{m}}_k)(\mathbf{x}_i - \bar{\mathbf{m}}_k)^T. \tag{1}$$

Unlike LDA which uses the average of the distances between each class mean and the sample mean as the between-class scatter measure, here we use the minimum of the pairwise distances between class means as the between-class scatter measure:

$$\rho_b = \min_{i,j} \left\{ \operatorname{tr}(\mathbf{W}^T (\bar{\mathbf{m}}_i - \bar{\mathbf{m}}_j)(\bar{\mathbf{m}}_i - \bar{\mathbf{m}}_j)^T \mathbf{W}) \right\}. \tag{2}$$

Also, we define the new within-class scatter measure as

$$\rho_w = \max_{i} \left\{ \operatorname{tr}(\mathbf{W}^T \mathbf{S}_i \mathbf{W}) \right\}, \tag{3}$$

which is the maximum of the average within-class pairwise distances.

Similar to LDA, we define the optimality criterion of WLDA as the ratio of the between-class scatter measure to the within-class scatter measure:

$$\max_{\mathbf{W}} \qquad J(\mathbf{W}) = \frac{\rho_b}{\rho_w}$$

$$\text{s.t.} \qquad \mathbf{W}^T\mathbf{W} = \mathbf{I}_r, \tag{4}$$

where $\mathbf{I}_r$ denotes the $r \times r$ identity matrix. The orthonormality constraint in problem (4) is widely used by many existing dimensionality reduction methods. Its role is to limit the scale of each column of $\mathbf{W}$ and eliminate the redundancy among all columns of $\mathbf{W}$.

## 2.2 Optimization Procedure

Since problem (4) is not easy to optimize with respect to $\mathbf{W}$, we resort to formulate this dimensionality reduction problem as a metric learning problem [22, 21, 4]. We define a new variable $\mathbf{\Sigma} = \mathbf{W}\mathbf{W}^T$ which can be used to define a metric. Then we express $\rho_b$ and $\rho_w$ in terms of $\mathbf{\Sigma}$ as

$$\rho_b = \min_{i,j}\left\{ \text{tr}\big((\bar{\mathbf{m}}_i - \bar{\mathbf{m}}_j)(\bar{\mathbf{m}}_i - \bar{\mathbf{m}}_j)^T\mathbf{\Sigma}\big)\right\}$$

$$\rho_w = \max_{i}\left\{ \text{tr}\big(\mathbf{S}_i\mathbf{\Sigma}\big)\right\},$$

due to a property of the matrix trace that $\text{tr}(\mathbf{AB}) = \text{tr}(\mathbf{BA})$ for any matrices $\mathbf{A}$ and $\mathbf{B}$ with proper sizes. The orthonormality constraint in problem (4) is non-convex with respect to $\mathbf{W}$ and cannot be expressed in terms of $\mathbf{\Sigma}$.

We define a set $\mathcal{M}_w$ as

$$\mathcal{M}_w = \left\{\mathbf{M}_w \mid \mathbf{M}_w = \mathbf{W}\mathbf{W}^T, \mathbf{W}^T\mathbf{W} = \mathbf{I}_r, \mathbf{W} \in \mathbb{R}^{d \times r}\right\}.$$

Apparently $\mathbf{\Sigma} \in \mathcal{M}_w$. It has been shown in [16] that the convex hull of $\mathcal{M}_w$ can be precisely expressed as a convex set $\mathcal{M}_e$ given by

$$\mathcal{M}_e = \left\{\mathbf{M}_e \mid \text{tr}(\mathbf{M}_e) = r, \mathbf{0} \preceq \mathbf{M}_e \preceq \mathbf{I}_d\right\},$$

where $\mathbf{0}$ denotes the zero vector or matrix of appropriate size and $\mathbf{A} \preceq \mathbf{B}$ means that the matrix $\mathbf{B} - \mathbf{A}$ is positive semidefinite. Each element in $\mathcal{M}_w$ is referred to as an extreme point of $\mathcal{M}_e$. Since $\mathcal{M}_e$ consists of all convex combinations of the elements in $\mathcal{M}_w$, $\mathcal{M}_e$ is the smallest convex set that contains $\mathcal{M}_w$, and hence $\mathcal{M}_w \subseteq \mathcal{M}_e$. Then problem (4) can be relaxed as

$$\max_{\mathbf{\Sigma}} \qquad J(\mathbf{\Sigma}) = \frac{\min_{i,j}\left\{\text{tr}\big(\mathbf{S}_{ij}\mathbf{\Sigma}\big)\right\}}{\max_{i}\left\{\text{tr}\big(\mathbf{S}_i\mathbf{\Sigma}\big)\right\}}$$

$$\text{s.t.} \qquad \text{tr}(\mathbf{\Sigma}) = r, \mathbf{0} \preceq \mathbf{\Sigma} \preceq \mathbf{I}_d, \tag{5}$$

where $\mathbf{S}_{ij} = (\bar{\mathbf{m}}_i - \bar{\mathbf{m}}_j)(\bar{\mathbf{m}}_i - \bar{\mathbf{m}}_j)^T$. For notational simplicity, we denote the constraint set as $\mathbb{C} = \{\mathbf{\Sigma} \mid \text{tr}(\mathbf{\Sigma}) = r, \mathbf{0} \preceq \mathbf{\Sigma} \preceq \mathbf{I}_d\}$. Table 1 shows an iterative algorithm for solving problem (5).

Table 1: Algorithm for solving optimization problem (5)

| |
| --- |
| Input: $\{\bar{\mathbf{m}}_i\}$, $\{\mathbf{S}_i\}$ and $r$ |
| 1: Initialize $\mathbf{\Sigma}^{(0)}$; <br> 2: For $k = 1, \dots, N_{iter}$ <br>     2.1: Compute the ratio $\alpha_k$ from $\mathbf{\Sigma}^{(k-1)}$ as: $\alpha_k = J(\mathbf{\Sigma}^{(k-1)})$; <br>     2.2: Solve the optimization problem <br>         $\mathbf{\Sigma}^{(k)} = \arg\max_{\mathbf{\Sigma} \in \mathbb{C}} \min_{i,j}\left\{\text{tr}\big(\mathbf{S}_{ij}\mathbf{\Sigma}\big)\right\} - \alpha_k\max_i\left\{\text{tr}\big(\mathbf{S}_i\mathbf{\Sigma}\big)\right\}$; <br>     2.3: If $\|\mathbf{\Sigma}^{(k)} - \mathbf{\Sigma}^{(k-1)}\|_F \leq \varepsilon$ (here we set $\varepsilon = 10^{-4}$) <br>         break; |
| Output: $\mathbf{\Sigma}$ |

We now present the solution of the optimization problem in step 2.2. It is equivalent to the following problem

$$\min_{\mathbf{\Sigma} \in \mathbb{C}} \alpha_k\max_{i}\left\{\text{tr}\big(\mathbf{S}_i\mathbf{\Sigma}\big)\right\} - \min_{i,j}\left\{\text{tr}\big(\mathbf{S}_{ij}\mathbf{\Sigma}\big)\right\}. \tag{6}$$

According to [3], we know that $\max_i \left\{ \mathrm{tr}\big(\mathbf{S}_i \boldsymbol{\Sigma}\big) \right\}$ is a convex function because it is the maximum of several convex functions, and $\min_{i,j} \left\{ \mathrm{tr}\big(\mathbf{S}_{ij} \boldsymbol{\Sigma}\big) \right\}$ is a concave function because it is the minimum of several concave functions. Moreover, $\alpha_k$ is a positive scalar since $\alpha_k = J(\boldsymbol{\Sigma}^{(k-1)})$. So problem (6) is a convex optimization problem. We introduce new variables $s$ and $t$ to simplify problem (6) as

$$
\begin{aligned}
\min_{\boldsymbol{\Sigma}, s, t} \quad & \alpha_k s - t \\
\text{s.t.} \quad & \mathrm{tr}\big(\mathbf{S}_i \boldsymbol{\Sigma}\big) \le s, \ \forall i \\
& \mathrm{tr}\big(\mathbf{S}_{ij} \boldsymbol{\Sigma}\big) \ge t > 0, \ \forall i, j \\
& \mathrm{tr}(\boldsymbol{\Sigma}) = r, \ \mathbf{0} \preceq \boldsymbol{\Sigma} \preceq \mathbf{I}_d.
\end{aligned} \tag{7}
$$

Note that problem (7) is a semidefinite programming (SDP) problem [19] which can be solved using a standard SDP solver. After obtaining the optimal $\boldsymbol{\Sigma}^\star$, we can recover the optimal $\mathbf{W}^\star$ as the top $r$ eigenvectors of $\boldsymbol{\Sigma}^\star$. In the following, we will prove the convergence of the algorithm in Table 1.

**Theorem 1** *For the algorithm in Table 1, we have* $J(\boldsymbol{\Sigma}^{(k)}) \ge J(\boldsymbol{\Sigma}^{(k-1)})$.

**Proof:** We define $g(\boldsymbol{\Sigma}) = \min_{i,j} \left\{ \mathrm{tr}\big(\mathbf{S}_{ij} \boldsymbol{\Sigma}\big) \right\} - \alpha_k \max_i \left\{ \mathrm{tr}\big(\mathbf{S}_i \boldsymbol{\Sigma}\big) \right\}$. Then $g(\boldsymbol{\Sigma}^{(k-1)}) = 0$ since $\alpha_k = \dfrac{\min_{i,j} \left\{ \mathrm{tr}\big(\mathbf{S}_{ij} \boldsymbol{\Sigma}^{(k-1)}\big) \right\}}{\max_i \left\{ \mathrm{tr}\big(\mathbf{S}_i \boldsymbol{\Sigma}^{(k-1)}\big) \right\}}$. Because $\boldsymbol{\Sigma}^{(k)} = \arg\max_{\boldsymbol{\Sigma} \in \mathbb{C}} g(\boldsymbol{\Sigma})$ and $\boldsymbol{\Sigma}^{(k-1)} \in \mathbb{C}$, we have $g(\boldsymbol{\Sigma}^{(k)}) \ge g(\boldsymbol{\Sigma}^{(k-1)}) = 0$. This means

$$
\frac{\min_{i,j} \left\{ \mathrm{tr}\big(\mathbf{S}_{ij} \boldsymbol{\Sigma}^{(k)}\big) \right\}}{\max_i \left\{ \mathrm{tr}\big(\mathbf{S}_i \boldsymbol{\Sigma}^{(k)}\big) \right\}} \ge \alpha_k,
$$

which implies that $J(\boldsymbol{\Sigma}^{(k)}) \ge J(\boldsymbol{\Sigma}^{(k-1)})$. $\qquad\square$

**Theorem 2** *For any $\boldsymbol{\Sigma} \in \mathbb{C}$, we have $0 \le J(\boldsymbol{\Sigma}) \le \frac{2\mathrm{tr}(\mathbf{S}_b)}{\sum_{i=1}^r \lambda_{d-i+1}}$ where $\lambda_i$ is the $i$th largest eigenvalue of $\mathbf{S}_w$.*

**Proof:** It is obvious that $J(\boldsymbol{\Sigma}) \ge 0$. The numerator of $J(\boldsymbol{\Sigma})$ can be upper-bounded as

$$
\min_{i,j} \left\{ \mathrm{tr}\big(\mathbf{S}_{ij} \boldsymbol{\Sigma}\big) \right\} \le \frac{\sum_{i=1}^C \sum_{j=1}^C n_i n_j \, \mathrm{tr}\big(\mathbf{S}_{ij} \boldsymbol{\Sigma}\big)}{\sum_{i=1}^C \sum_{j=1}^C n_i n_j} = 2\mathrm{tr}(\mathbf{S}_b \boldsymbol{\Sigma}) \le 2\mathrm{tr}(\mathbf{S}_b). \tag{8}
$$

Moreover, the denominator of $J(\boldsymbol{\Sigma})$ can be lower-bounded as

$$
\max_i \left\{ \mathrm{tr}\big(\mathbf{S}_i \boldsymbol{\Sigma}\big) \right\} \ge \frac{\sum_{i=1}^C n_i \, \mathrm{tr}\big(\mathbf{S}_i \boldsymbol{\Sigma}\big)}{\sum_{i=1}^C n_i} = \mathrm{tr}\big(\mathbf{S}_w \boldsymbol{\Sigma}\big) \ge \sum_{i=1}^d \lambda_{d-i+1} \tilde{\lambda}_i \ge \sum_{i=1}^r \lambda_{d-i+1}, \tag{9}
$$

where $\tilde{\lambda}_i$ is the $i$th largest eigenvalue of $\boldsymbol{\Sigma}$ and satisfies $0 \le \tilde{\lambda}_i \le 1$ and $\sum_{i=1}^d \tilde{\lambda}_i = r$ due to the constraints $\mathbb{C}$ on $\boldsymbol{\Sigma}$. By utilizing Eqs. (8) and (9), we can reach the conclusion. $\qquad\square$

From Theorem 2, we can see that $J(\boldsymbol{\Sigma})$ is bounded and our method is non-decreasing. So our method can achieve a local optimum when converged.

### 2.3   Optimization in Dual Form

In the previous subsection, we need to solve the SDP problem in problem (7). However, SDP is not scalable to high dimensionality $d$. In many real-world applications to which dimensionality reduction is applied, the number of data points $n$ is much smaller than the dimensionality $d$. Under such circumstances, speedup can be obtained by solving the dual form of problem (4) instead.

It is easy to show that the solution of problem (4) satisfies $\mathbf{W} = \mathbf{X}\mathbf{A}$ [14] where $\mathbf{X} = (\mathbf{x}_1, \ldots, \mathbf{x}_n)$ is the data matrix and $\mathbf{A} \in \mathbb{R}^{n \times r}$. Then problem (4) can be formulated as

$$
\begin{aligned}
\max_{\mathbf{A}} \quad & \frac{\min_{i,j} \left\{ \mathrm{tr}\big(\mathbf{A}^T \mathbf{X}^T \mathbf{S}_{ij} \mathbf{X} \mathbf{A}\big) \right\}}{\max_i \left\{ \mathrm{tr}\big(\mathbf{A}^T \mathbf{X}^T \mathbf{S}_i \mathbf{X} \mathbf{A}\big) \right\}} \\
\text{s.t.} \quad & \mathbf{A}^T \mathbf{K} \mathbf{A} = \mathbf{I}_r,
\end{aligned} \tag{10}
$$

where $\mathbf{K} = \mathbf{X}^T\mathbf{X}$ is the linear kernel matrix. Here we assume that $\mathbf{K}$ is positive definite since the data points are independent and identically distributed and $d$ is much larger than $n$. We define a new variable $\mathbf{B} = \mathbf{K}^{\frac{1}{2}}\mathbf{A}$ and problem (10) can be reformulated as

$$\max_{\mathbf{B}} \quad \frac{\min_{i,j}\left\{\operatorname{tr}\left(\mathbf{B}^T\mathbf{K}^{-\frac{1}{2}}\mathbf{X}^T\mathbf{S}_{ij}\mathbf{X}\mathbf{K}^{-\frac{1}{2}}\mathbf{B}\right)\right\}}{\max_i\left\{\operatorname{tr}\left(\mathbf{B}^T\mathbf{K}^{-\frac{1}{2}}\mathbf{X}^T\mathbf{S}_i\mathbf{X}\mathbf{K}^{-\frac{1}{2}}\mathbf{B}\right)\right\}}$$

$$\text{s.t.} \quad \mathbf{B}^T\mathbf{B} = \mathbf{I}_r. \tag{11}$$

Note that problem (11) is almost the same as problem (4) and so we can use the same relaxation technique above to solve problem (11). In the relaxed problem, the variable $\tilde{\mathbf{\Sigma}} = \mathbf{B}\mathbf{B}^T$ used to define the metric in the dual form is of size $n \times n$ which is much smaller than that $(d \times d)$ of $\mathbf{\Sigma}$ in the primal form when $n < d$. So solving the problem in the dual form is more efficient. Moreover, the dual form facilitates kernel extension of our method.

## 2.4  Alternative Optimization Procedure

In case the number of training data points $n$ and the dimensionality $d$ are both large, the above optimization procedures will be infeasible. Here we introduce yet another optimization procedure based on a greedy approach to solve problem (4) when both $n$ and $d$ are large.

We find the first column of $\mathbf{W}$ by solving problem (4) where $\mathbf{W}$ is a vector, then find the second column of $\mathbf{W}$ by assuming the first column is fixed, and so on. This procedure consists of $r$ steps. In the $k$th step, we assume that the first $k-1$ columns of $\mathbf{W}$ have been obtained and we find the $k$th column according to problem (4). We use $\mathbf{W}_{k-1}$ to denote the matrix in which the first $k-1$ columns are already known and the constraint in problem (4) becomes

$$\mathbf{w}_k^T\mathbf{w}_k = 1, \ \mathbf{W}_{k-1}^T\mathbf{w}_k = \mathbf{0}.$$

When $k=1$, $\mathbf{W}_{k-1}^T$ can be viewed as an empty matrix and the constraint $\mathbf{W}_{k-1}^T\mathbf{w}_k = \mathbf{0}$ does not exist. So in the $k$th step, we need to solve the following problem

$$\min_{\mathbf{w}_k, s, t} \quad \frac{s}{t}$$

$$\text{s.t.} \quad \mathbf{w}_k^T\mathbf{S}_i\mathbf{w}_k + a_i - s \le 0, \ \forall i$$

$$t - \mathbf{w}_k^T(\bar{\mathbf{m}}_i - \bar{\mathbf{m}}_j)(\bar{\mathbf{m}}_i - \bar{\mathbf{m}}_j)^T\mathbf{w}_k - b_{ij} \le 0, \ \forall i, j$$

$$s, t > 0$$

$$\mathbf{w}_k^T\mathbf{w}_k \le 1, \ \mathbf{W}_{k-1}^T\mathbf{w}_k = \mathbf{0}, \tag{12}$$

where $a_i = \operatorname{tr}\left(\mathbf{W}_{k-1}^T\mathbf{S}_i\mathbf{W}_{k-1}\right)$ and $b_{ij} = \operatorname{tr}\left(\mathbf{W}_{k-1}^T(\bar{\mathbf{m}}_i - \bar{\mathbf{m}}_j)(\bar{\mathbf{m}}_i - \bar{\mathbf{m}}_j)^T\mathbf{W}_{k-1}\right)$. In the last constraint of problem (12), we relax the constraint on $\mathbf{w}_k$ as $\mathbf{w}_k^T\mathbf{w}_k \le 1$ to make it convex.

The function $\frac{s}{t}$ is not convex with respect to $(s,t)^T$ since the Hessian matrix is not positive semidefinite. So the objective function of problem (12) is non-convex. Moreover, the second constraint in problem (12), which is the difference of two convex functions, is also non-convex. We rewrite the objective function as

$$\frac{s}{t} = \frac{(s+1)^2}{4t} - \frac{(s-1)^2}{4t},$$

which is also the difference of two convex functions since $f(x,y) = \frac{(x+b)^2}{y}$ for $y > 0$ is convex with respect to $x$ and $y$ according to [3]. Then we can use the constrained concave-convex procedure (CCCP) [24, 18] to optimize problem (12). More specifically, in the $(l+1)$th iteration of CCCP, we replace the non-convex parts of the objective function and the second constraint with their first-order Taylor expansions at the solution $\{s^{(l)}, t^{(l)}, \mathbf{w}_k^{(l)}\}$ in the $l$th iteration and solve the following problem

$$\min_{\mathbf{w}_k, s, t} \quad \frac{(s+1)^2}{4t} - cs + c^2t$$

$$\text{s.t.} \quad \mathbf{w}_k^T\mathbf{S}_i\mathbf{w}_k + a_i - s \le 0, \ \forall i$$

$$t - 2(\mathbf{w}_k^{(l)})^T(\bar{\mathbf{m}}_i - \bar{\mathbf{m}}_j)(\bar{\mathbf{m}}_i - \bar{\mathbf{m}}_j)^T\mathbf{w}_k + h_{ij}^{(l)} - b_{ij} \le 0, \ \forall i, j$$

$$s, t > 0$$

$$\mathbf{w}_k^T\mathbf{w}_k \le 1, \ \mathbf{W}_{k-1}^T\mathbf{w}_k = \mathbf{0}, \tag{13}$$

where $c = \frac{s^{(l)}-1}{2t^{(l)}}$ and $h_{ij}^{(l)} = (\mathbf{w}_k^{(l)})^T(\bar{\mathbf{m}}_i - \bar{\mathbf{m}}_j)(\bar{\mathbf{m}}_i - \bar{\mathbf{m}}_j)^T\mathbf{w}_k^{(l)}$. By putting an upper bound on $\frac{(s+1)^2}{4t}$, i.e., $\frac{(s+1)^2}{4t} \leq u$, and using the fact that

$$\frac{(s+1)^2}{4t} \leq u \ (u, t > 0) \ \Leftrightarrow \ \left\|\begin{matrix} s+1 \\ u-t \end{matrix}\right\|_2 \leq u+t,$$

where $\|\cdot\|_2$ denotes the 2-norm of a vector, we can reformulate problem (13) into a second-order cone programming (SOCP) problem [12] which is more efficient than SDP:

$$
\begin{aligned}
\min_{\mathbf{w}_k, u, s, t} \quad & u - cs + c^2 t \\
\text{s.t.} \quad & \mathbf{w}_k^T \mathbf{S}_i \mathbf{w}_k + a_i - s \leq 0, \ \forall i \\
& t - 2(\mathbf{w}_k^{(l)})^T(\bar{\mathbf{m}}_i - \bar{\mathbf{m}}_j)(\bar{\mathbf{m}}_i - \bar{\mathbf{m}}_j)^T\mathbf{w}_k + h_{ij}^{(l)} - b_{ij} \leq 0, \ \forall i, j \\
& \left\|\begin{matrix} s+1 \\ u-t \end{matrix}\right\|_2 \leq u + t \text{ with } u, s, t > 0 \\
& \mathbf{w}_k^T \mathbf{w}_k \leq 1, \ \mathbf{W}_{k-1}^T \mathbf{w}_k = \mathbf{0}.
\end{aligned}
\tag{14}
$$

## 2.5 Analysis

It is well known that in binary classification problems when both classes are normally distributed with the same covariance matrix, the solution given by conventional LDA is the Bayes optimal solution. We will show here that this property still holds for WLDA.

The objective function for WLDA in a binary classification problem is formulated as

$$
\begin{aligned}
\max_{\mathbf{w}} \quad & \frac{\mathbf{w}^T(\bar{\mathbf{m}}_1 - \bar{\mathbf{m}}_2)(\bar{\mathbf{m}}_1 - \bar{\mathbf{m}}_2)^T\mathbf{w}}{\max\{\mathbf{w}^T\mathbf{S}_1\mathbf{w}, \mathbf{w}^T\mathbf{S}_2\mathbf{w}\}} \\
\text{s.t.} \quad & \mathbf{w} \in \mathbb{R}^d, \mathbf{w}^T\mathbf{w} \leq 1.
\end{aligned}
\tag{15}
$$

Here, similar to conventional LDA, the reduced dimensionality $r$ is set to 1. When the two classes have the same covariance matrix, i.e., $\mathbf{S}_1 = \mathbf{S}_2$, the problem degenerates to the optimization problem of conventional LDA since $\mathbf{w}^T\mathbf{S}_1\mathbf{w} = \mathbf{w}^T\mathbf{S}_2\mathbf{w}$ for any $\mathbf{w}$ and $\mathbf{w}$ is the solution of conventional LDA.[1] So WLDA also gives the same Bayes optimal solution as conventional LDA.

Since the scale of $\mathbf{w}$ does not affect the final solution in problem (15), we simplify problem (15) as

$$
\begin{aligned}
\max_{\mathbf{w}} \quad & \mathbf{w}^T(\bar{\mathbf{m}}_1 - \bar{\mathbf{m}}_2)(\bar{\mathbf{m}}_1 - \bar{\mathbf{m}}_2)^T\mathbf{w} \\
\text{s.t.} \quad & \mathbf{w}^T\mathbf{S}_1\mathbf{w} \leq 1, \ \mathbf{w}^T\mathbf{S}_2\mathbf{w} \leq 1.
\end{aligned}
\tag{16}
$$

Since problem (16) is to maximize a convex function, it is not a convex problem. We can still use CCCP to optimize problem (16). In the $(l+1)$th iteration of CCCP, we need to solve the following problem

$$
\begin{aligned}
\max_{\mathbf{w}} \quad & (\mathbf{w}^{(l)})^T(\bar{\mathbf{m}}_1 - \bar{\mathbf{m}}_2)(\bar{\mathbf{m}}_1 - \bar{\mathbf{m}}_2)^T\mathbf{w} \\
\text{s.t.} \quad & \mathbf{w}^T\mathbf{S}_1\mathbf{w} \leq 1, \ \mathbf{w}^T\mathbf{S}_2\mathbf{w} \leq 1.
\end{aligned}
\tag{17}
$$

The Lagrangian is given by

$$L = -(\mathbf{w}^{(l)})^T(\bar{\mathbf{m}}_1 - \bar{\mathbf{m}}_2)(\bar{\mathbf{m}}_1 - \bar{\mathbf{m}}_2)^T\mathbf{w} + \alpha(\mathbf{w}^T\mathbf{S}_1\mathbf{w} - 1) + \beta(\mathbf{w}^T\mathbf{S}_2\mathbf{w} - 1),$$

where $\alpha \geq 0$ and $\beta \geq 0$. We calculate the gradients of $L$ with respect to $\mathbf{w}$ and set them to 0 to obtain

$$\mathbf{w} = (2\alpha\mathbf{S}_1 + 2\beta\mathbf{S}_2)^{-1}(\bar{\mathbf{m}}_1 - \bar{\mathbf{m}}_2)(\bar{\mathbf{m}}_1 - \bar{\mathbf{m}}_2)^T\mathbf{w}^{(l)}.$$

From this, we can see that when the algorithm converges, the optimal $\mathbf{w}^\star$ satisfies

$$\mathbf{w}^\star \propto (2\alpha^\star\mathbf{S}_1 + 2\beta^\star\mathbf{S}_2)^{-1}(\bar{\mathbf{m}}_1 - \bar{\mathbf{m}}_2).$$

This is similar to the following property of the optimal solution in conventional LDA

$$\mathbf{w}^\star \propto \mathbf{S}_w^{-1}(\bar{\mathbf{m}}_1 - \bar{\mathbf{m}}_2) \propto (n_1\mathbf{S}_1 + n_2\mathbf{S}_2)^{-1}(\bar{\mathbf{m}}_1 - \bar{\mathbf{m}}_2).$$

However, in our method, $\alpha^\star$ and $\beta^\star$ are not fixed but learned from the following dual problem

$$\min_{\alpha,\beta} \quad \frac{\gamma}{4}(\bar{\mathbf{m}}_1 - \bar{\mathbf{m}}_2)(\alpha\mathbf{S}_1 + \beta\mathbf{S}_2)^{-1}(\bar{\mathbf{m}}_1 - \bar{\mathbf{m}}_2) + \alpha + \beta$$
$$\text{s.t.} \quad \alpha \geq 0, \ \beta \geq 0, \tag{18}$$

where $\gamma = \left((\bar{\mathbf{m}}_1 - \bar{\mathbf{m}}_2)^T \mathbf{w}^{(l)}\right)^2$. Note that the first term in the objective function of problem (18) is just the scaled optimality criterion of conventional LDA when we assume the within-class scatter matrix $\mathbf{S}_w$ to be $\mathbf{S}_w = \alpha\mathbf{S}_1 + \beta\mathbf{S}_2$. From this view, WLDA seeks to find a linear combination of $\mathbf{S}_1$ and $\mathbf{S}_2$ as the within-class scatter matrix to maximize the optimality criterion of conventional LDA while controlling the complexity of the within-class scatter matrix as reflected by the second and third terms of the objective function in problem (18).

## 3 Related Work

In [11], Li et al. proposed a maximum margin criterion for dimensionality reduction by changing the optimization problem of conventional LDA to: $\max_{\mathbf{W}} \ \text{tr}\left(\mathbf{W}^T(\mathbf{S}_b - \mathbf{S}_w)\mathbf{W}\right)$. The objective function has a physical meaning similar to that of LDA which favors a large between-class scatter measure and a small within-class scatter measure. However, similar to LDA, the maximum margin criterion also uses the average distances to describe the between-class and within-class scatter measures. Kocsor et al. [10] proposed another maximum margin criterion for dimensionality reduction. The objective function in [10] is identical to that of support vector machine (SVM) and it treats the decision function in SVM as one direction in the transformation matrix $\mathbf{W}$.

In [9], Kim et al. proposed a robust LDA algorithm to deal with data uncertainty in classification applications by formulating the problem as a convex problem. However, in many applications, it is not easy to obtain the information about data uncertainty. Moreover, its limitation is that it can only handle binary classification problems but not more general multi-class problems.

The orthogonality constraint on the transformation matrix $\mathbf{W}$ has been widely used by dimensionality reduction methods, such as Foley-Sammon LDA (FSLDA) [6, 5] and orthogonal LDA [23]. The orthogonality constraint can help to eliminate the redundant information in $\mathbf{W}$. This has been shown to be effective for dimensionality reduction.

## 4 Experimental Validation

In this section, we evaluate WLDA empirically on some benchmark datasets and compare WLDA with several related methods, including conventional LDA, trace-ratio LDA [20], FSLDA [6, 5], and MarginLDA [11]. For fair comparison with conventional LDA, we set the reduced dimensionality of each method compared to $C - 1$ where $C$ is the number of classes in the dataset. After dimensionality reduction, we use a simple nearest-neighbor classifier to perform classification. Our choice of the optimization procedure follows this strategy: when the number of features $d$ or the number of training data points $n$ is smaller than 100, the optimization method in Section 2.2 or 2.3 is used depending on which one is smaller; otherwise, we use the greedy method in Section 2.4.

### 4.1 Experiments on UCI Datasets

Ten UCI datasets [1] are used in the first set of experiments. For each dataset, we randomly select 70% to form the training set and the rest for the test set. We perform 10 random splits and report in Table 2 the average results across the 10 trials. For each setting, the lowest classification error is shown in bold. We can see that WLDA gives the best result for most datasets. For some datasets, e.g., balance-scale and hayes-roth, even though WLDA is not the best, the difference between it and the best one is very small. Thus it is fair to say that the results obtained demonstrate convincingly the effectiveness of WLDA.

### 4.2 Experiments on Face and Object Datasets

Dimensionality reduction methods have been widely used for face and object recognition applications. Previous research found that face and object images usually lie in a low-dimensional subspace

Table 2: Average classification errors on the UCI datasets. Here tr-LDA denotes the trace-ratio LDA [20].

| Dataset | LDA | tr-LDA | FSLDA | MarginLDA | WLDA |
|---|---|---|---|---|---|
| diabetes | 0.3233 | 0.3143 | 0.4039 | 0.4143 | **0.2996** |
| heart | 0.2448 | 0.2259 | 0.4395 | 0.2407 | **0.2157** |
| liver | 0.4001 | 0.3933 | 0.4365 | 0.5058 | **0.3779** |
| sonar | 0.2806 | 0.2895 | 0.3694 | 0.2806 | **0.2661** |
| spambase | 0.1279 | 0.1301 | 0.3093 | 0.1440 | **0.1260** |
| balance-scale | 0.1193 | 0.1198 | 0.1176 | **0.1150** | 0.1174 |
| iris | 0.0244 | 0.0267 | 0.0622 | 0.0644 | **0.0211** |
| hayes-roth | 0.3125 | 0.3104 | 0.3104 | **0.2958** | 0.3050 |
| waveform | 0.1861 | 0.1865 | 0.2261 | 0.2303 | **0.1671** |
| mfeat-factors | 0.0732 | 0.0518 | 0.0868 | 0.0817 | **0.0250** |

of the ambient image space. Fisherface (based on LDA) [2] is one representative dimensionality reduction method. We use three face databases, ORL [2], PIE [17] and AR [13], and one object database, COIL [15], in our experiments. In the AR face database, 2,600 images of 100 persons (50 men and 50 women) are used. Before the experiment, each image is converted to gray scale and normalized to a size of $33 \times 24$ pixels. The ORL face database contains 400 face images of 40 persons, each having 10 images. Each image is preprocessed to a size of $28 \times 23$ pixels. In our experiment, we choose the frontal pose from the PIE database with varying lighting and illumination conditions. There are about 49 images for each subject. Before the experiment, we resize each image to a resolution of $32 \times 32$ pixels. The COIL database contains 1,440 grayscale images with black background for 20 objects with each object having 72 different images.

In face and object recognition applications, the size of the training set is usually not very large since labeling data is very laborious and costly. To simulate this realistic situation, we randomly choose 4 images of a person or object in the database to form the training set and the remaining images to form the test set. We perform 10 random splits and report the average classification error rates across the 10 trials in Table 3. From the result, we can see that WLDA is comparable to or even better than the other methods compared.

Table 3: Average classification errors on the face and object datasets. Here tr-LDA denotes the trace-ratio LDA [20].

| Dataset | LDA | tr-LDA | FSLDA | MarginLDA | WLDA |
|---|---|---|---|---|---|
| ORL | 0.1529 | 0.1042 | 0.0654 | 0.0536 | **0.0446** |
| PIE | 0.4305 | 0.2527 | 0.6715 | 0.2936 | **0.2469** |
| AR | 0.2498 | **0.1919** | 0.7726 | 0.4282 | 0.1965 |
| COIL | 0.2554 | 0.1737 | 0.1726 | 0.1653 | **0.1593** |

## 5 Conclusion

In this paper, we have presented a new supervised dimensionality reduction method by exploiting the worst-case view instead of average-case view in the formulation. One interesting direction of our future work is to extend WLDA to handle tensors for 2D or higher-order data. Moreover, we will investigate the semi-supervised extension of WLDA to exploit the useful information contained in the unlabeled data available in some applications.

## Acknowledgement

This research has been supported by General Research Fund 621407 from the Research Grants Council of Hong Kong.

## Footnotes

[1]The constraint $\mathbf{w}^T\mathbf{w} \leq 1$ in problem (15) only serves to limit the scale of $\mathbf{w}$.

# References

[1] A. Asuncion and D.J. Newman. UCI machine learning repository, 2007.

[2] P. N. Belhumeur, J. P. Hespanha, and D. J. Kriegman. Eigenfaces vs. Fisherfaces: Recognition using class specific linear projection. *IEEE Transactions on Pattern Analysis and Machine Intelligence*, 19(7):711–720, 1997.

[3] S. Boyd and L. Vandenberghe. *Convex Optimization*. Cambridge University Press, New York, NY, 2004.

[4] J. V. Davis, B. Kulis, P. Jain, S. Sra, and I. S. Dhillon. Information-theoretic metric learning. In *Proceedings of the Twenty-Fourth International Conference on Machine Learning*, pages 209–216, Corvalis, Oregon, USA, 2007.

[5] J. Duchene and S. Leclercq. An optimal transformation for discriminant and principal component analysis. *IEEE Transactions on Pattern Analysis and Machine Intelligence*, 10(6):978–983, 1988.

[6] D. H. Foley and J. W. Sammon. An optimal set of discriminant vectors. *IEEE Transactions on Computers*, 24(3):281–289, 1975.

[7] K Fukunnaga. *Introduction to Statistical Pattern Recognition*. Academic Press, New York, 1991.

[8] I. T. Jolliffe. *Principal Component Analysis*. Springer-Verlag, New York, 2nd edition, 2002.

[9] S.-J. Kim, A. Magnani, and S. Boyd. Robust Fisher discriminant analysis. In Y. Weiss, B. Schölkopf, and J. Platt, editors, *Advances in Neural Information Processing Systems 18*, pages 659–666. Vancouver, British Columbia, Canada, 2006.

[10] A. Kocsor, K. Kovács, and C. Szepesvári. Margin maximizing discriminant analysis. In *Proceedings of the 15th European Conference on Machine Learning*, pages 227–238, Pisa, Italy, 2004.

[11] H. Li, T. Jiang, and K. Zhang. Efficient and robust feature extraction by maximum margin criterion. In S. Thrun, L. K. Saul, and B. Schölkopf, editors, *Advances in Neural Information Processing Systems 16*, Vancouver, British Columbia, Canada, 2003.

[12] M. S. Lobo, L. Vandenberghe, S. Boyd, and H. Lebret. Applications of second-order cone programming. *Linear Algebra and its Applications*, 284:193–228, 1998.

[13] A. M. Martínez and R. Benavente. The AR-face database. Technical Report 24, CVC, 1998.

[14] S. Mika, G. Rätsch, J. Weston, B. Schölkopf, A. J. Smola, and K.-R. Müller. Constructing descriptive and discriminative nonlinear features: Rayleigh coefficients in kernel feature spaces. *IEEE Transactions on Pattern Analysis and Machine Intelligence*, 25(5):623–633, 2003.

[15] S. A. Nene, S. K. Nayar, and H. Murase. Columbia object image library (COIL-20). Technical Report 005, CUCS, 1996.

[16] M. L. Overton and R. S. Womersley. Optimality conditions and duality theory for minimizing sums of the largest eigenvalues of symmetric matrices. *Math Programming*, 62(2):321–357, 1993.

[17] T. Sim, S. Baker, and M. Bsat. The CMU pose, illumination and expression database. *IEEE Transactions on Pattern Analysis and Machine Intelligence*, 25(12):1615–1618, 2003.

[18] A. J. Smola, S. V. N. Vishwanathan, and T. Hofmann. Kernel methods for missing variables. In *Proceedings of the Tenth International Workshop on Artificial Intelligence and Statistics*, Barbados, 2005.

[19] L. Vandenberghe and S. Boyd. Semidefinite prgramming. *SIAM Review*, 38(1):49–95, 1996.

[20] H. Wang, S. Yan, D. Xu, X. Tang, and T. Huang. Trace ratio vs. ratio trace for dimensionality reduction. In *Proceedings of the IEEE Computer Society Conference on Computer Vision and Pattern Recognition*, pages 1–8, Minneapolis, Minnesota, USA, 2007.

[21] K. Q. Weinberger, J. Blitzer, and L. K. Saul. Distance metric learning for large margin nearest neighbor classification. In Y. Weiss, B. Schölkopf, and J. Platt, editors, *Advances in Neural Information Processing Systems 18*, pages 1473–1480, Vancouver, British Columbia, Canada, 2005.

[22] E. P. Xing, A. Y. Ng, M. I. Jordan, and S. J. Russell. Distance metric learning with application to clustering with side-information. In S. Becker, S. Thrun, and K. Obermayer, editors, *Advances in Neural Information Processing Systems 15*, pages 505–512, Vancouver, British Columbia, Canada, 2002.

[23] J.-P. Ye and T. Xiong. Computational and theoretical analysis of null space and orthogonal linear discriminant analysis. *Journal of Machine Learning Research*, 7:1183–1204, 2006.

[24] A. Yuille and A. Rangarajan. The concave-convex procedure. *Neural Computation*, 15(4):915–936, 2003.

